# Learning to localise sounds with spiking neural networks

**Dan F. M. Goodman**
Départment d'Etudes Cognitive
Ecole Normale Supérieure
29 Rue d'Ulm
Paris 75005, France
dan.goodman@ens.fr

**Romain Brette**
Départment d'Etudes Cognitive
Ecole Normale Supérieure
29 Rue d'Ulm
Paris 75005, France
romain.brette@ens.fr

## Abstract

To localise the source of a sound, we use location-specific properties of the signals received at the two ears caused by the asymmetric filtering of the original sound by our head and pinnae, the head-related transfer functions (HRTFs). These HRTFs change throughout an organism's lifetime, during development for example, and so the required neural circuitry cannot be entirely hardwired. Since HRTFs are not directly accessible from perceptual experience, they can only be inferred from filtered sounds. We present a spiking neural network model of sound localisation based on extracting location-specific synchrony patterns, and a simple supervised algorithm to learn the mapping between synchrony patterns and locations from a set of example sounds, with no previous knowledge of HRTFs. After learning, our model was able to accurately localise new sounds in both azimuth and elevation, including the difficult task of distinguishing sounds coming from the front and back.

Keywords: Auditory Perception & Modeling (Primary); Computational Neural Models, Neuroscience, Supervised Learning (Secondary)

## 1 Introduction

For many animals, it is vital to be able to quickly locate the source of an unexpected sound, for example to escape a predator or locate a prey. For humans, localisation cues are also used to isolate a speaker in a noisy environment. Psychophysical studies have shown that source localisation relies on a variety of acoustic cues such as interaural time and level differences (ITDs and ILDs) and spectral cues (Blauert 1997). These cues are highly dependent on the geometry of the head, body and pinnae, and can change significantly during an animal's lifetime, notably during its development but also in mature animals (which are known to be able to adapt to these changes, for example Hofman et al. 1998). Previous neural models addressed the mechanisms of cue extraction, in particular neural mechanisms underlying ITD sensitivity, using simplified binaural stimuli such as tones or noise bursts with artificially induced ITDs (Colburn, 1973; Reed and Blum, 1990; Gerstner et al., 1996; Harper and McAlpine, 2004; Zhou et al., 2005; Liu et al., 2008), but did not address the problem of learning to localise natural sounds in realistic acoustical environments.

Since the physical laws of sound propagation are linear, the sound $S$ produced by a source is received at any point $x$ of an acoustical environment as a linearly filtered version $F_x * S$ (linear convolution), where the filter is specific of the location $x$ of the listener, the location of the source and the acoustical environment (ground, wall, objects, etc.). For binaural hearing, the acoustical environment includes the head, body and pinnae, and the sounds received at the two ears are $F_L * S$ and $F_R * S$, where $(F_L, F_R)$ is a pair of location-specific filters. Because the two sounds originate from the same

signal, the binaural stimulus has a specific structure, which should result in synchrony patterns in the encoding neurons. Specifically, we modelled the response of monaural neurons by a linear filtering of the sound followed by a spiking nonlinearity. Two neurons $A$ and $B$ responding to two different sides (left and right), with receptive fields $N_A$ and $N_B$, transform the signals $N_A * F_L * S$ and $N_B * F_R * S$ into spike trains. Thus, synchrony between $A$ and $B$ occurs whenever $N_A * F_L = N_B * F_R$, i.e., for a specific set of filter pairs $(F_L, F_R)$. Thus, in our model, sounds presented at a given location induce specific synchrony patterns, which then activate a specific assembly of postsynaptic neurons (coincidence detection neurons), in a way that is independent of the source signal (see Goodman and Brette, in press). Learning a new location consists in assigning a label to the activated assembly, using a teacher signal (for example visual input).

We used measured human HRTFs to generate binaural signals at different source locations from a set of various sounds. These signals were used to train the model and we tested the localisation accuracy with new sounds. After learning, the model was able to accurately locate unknown sounds in both azimuth and elevation.

## 2 Methods

### 2.1 Virtual acoustics

Sound sources used were: broadband white noise; recordings of instruments and voices from the RWC Music Database (`http://staff.aist.go.jp/m.goto/RWC-MDB/`); and recordings of vowel-consonant-vowel sounds (Lorenzi et al., 1999). All sounds were of 1 second duration and were presented at 80 dB SPL. Sounds were filtered by head-related impulse responses (HRIRs) from the IRCAM LISTEN HRTF Database (`http://recherche.ircam.fr/equipes/salles/listen/index.html`). This database includes 187 approximately evenly spaced locations at all azimuths in 15 degree increments (except for high elevations) and elevations from -45 to 90 degrees in 15 degree increments. HRIRs from this and other databases do not provide sufficiently accurate timing information at frequencies below around 150Hz, and so subsequent cochlear filtering was restricted to frequencies above this point.

### 2.2 Mathematical principle

Consider two sets of neurons which respond monaurally to sounds from the left ear and from the right ear by filtering sounds through a linear filter $N$ (modeling their receptive field, corresponding to cochlear and neural transformations on the pathway between the ear and the neuron) followed by spiking. Each neuron has a different filter. Spiking is modeled by an integrate-and-fire description or some other spiking model. Consider two neurons $A$ and $B$ which respond to sounds from the left and right ear, respectively. When a sound $S$ is produced by a source at a given location, it arrives at the two ears as the binaural signal $(F_L * S, F_R * S)$ (convolution), where $(F_L, F_R)$ is the location-specific pair of acoustical filters. The filtered inputs to the two spiking models $A$ and $B$ are then $N_A * F_L * S$ and $N_B * F_R * S$. These will be identical for any sound $S$ whenever $N_A * F_L = N_B * F_R$, implying that the two neurons fire synchronously. For each location indicated by its filter pair $(F_L, F_R)$, we define the *synchrony pattern* as the set of binaural pairs of neurons $(A, B)$ such that $N_A * F_L = N_B * F_R$. This pattern is location-specific and independent of the source signal $S$. Therefore, the identity of the synchrony pattern induced by a binaural stimulus indicates the location of the source. Learning consists in assigning a synchrony pattern induced by a sound to the location of the source.

To have a better idea of these synchrony patterns, consider a pair of filters $(F_L^*, F_R^*)$ that corresponds to a particular location $x$ (azimuth, elevation, distance, and possibly also position of the listener in the acoustical environment), and suppose neuron $A$ has receptive field $N_A = F_R^*$ and neuron $B$ has receptive field $N_B = F_L^*$. Then neurons $A$ and $B$ fire in synchrony whenever $F_R^* * F_L = F_L^* * F_R$, in particular when $F_L = F_L^*$ and $F_R = F_R^*$, that is, at location $x$ (since convolution is commutative). More generally, if $U$ is a band-pass filter and the receptive fields of neurons $A$ and $B$ are $U * F_R^*$ and $U * F_L^*$, respectively, then the neurons fire synchronously at location $x$. The same property applies if a nonlinearity (e.g. compression) is applied after filtering. If the bandwidth of $U$ is very small, then $U * F_R^*$ is essentially the filter $U$ followed by a delay and gain. Therefore, to represent all possible

locations in pairs of neuron filters, we consider that the set of neural transformations $N$ is a bank of band-pass filters followed by a set of delays and gains.

To decode synchrony patterns, we define a set of binaural neurons which receive input spike trains from monaural neurons on both sides (two inputs per neuron). A binaural neuron responds preferentially when its two inputs are synchronous, so that synchrony patterns are mapped to assemblies of binaural neurons. Each location-specific assembly is the set of binaural neurons for which the input neurons fire synchronously at that location. This is conceptually similar to the Jeffress model (Jeffress, 1948), where a neuron is maximally activated when acoustical and axonal delays match, and related models (Lindemann, 1986; Gaik, 1993). However, the Jeffress model is restricted to azimuth estimation and it is difficult to implement it directly with neuron models because ILDs always co-occur with ITDs and disturb spike-timing.

## 2.3 Implementation with spiking neuron models

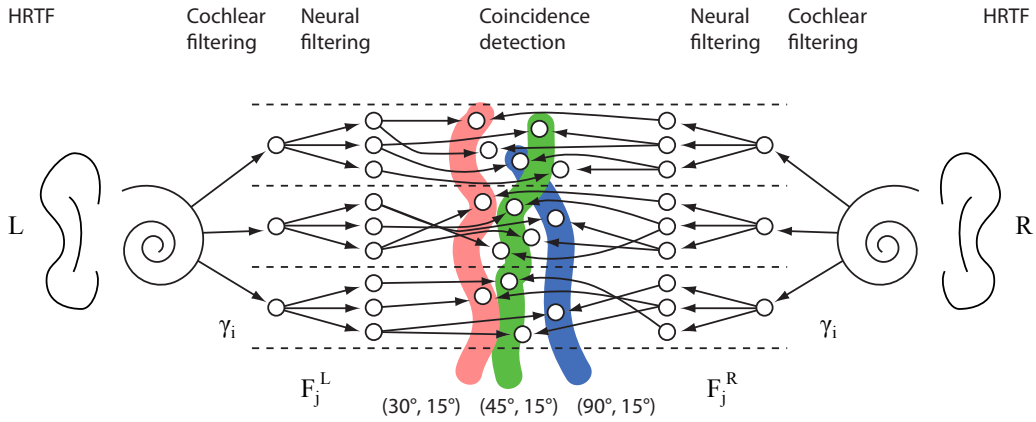

Figure 1: Implementation of the model. The source signal arrives at the two ears after acoustical filtering by HRTFs. The two monaural signals are filtered by a set of gammatone filters $\gamma_i$ with central frequencies between 150 Hz and 5 kHz (cochlear filtering). In each band (3 bands shown, between dashed lines), various gains and delays are applied to the signal (neural filtering $F_j^L$ and $F_j^R$) and spiking neuron models transform the resulting signals into spike trains, which converge from each side on a coincidence detector neuron (same neuron model). The neural assembly corresponding to a particular location is the set of coincidence detector neurons for which their input neurons fire in synchrony at that location (one pair for each frequency channel).

The overall structure and architecture of the model is illustrated in Figure 1. All programming was done in the Python programming language, using the "Brian" spiking neural network simulator package (Goodman and Brette, 2009). Simulations were performed on Intel i7 Core processors. The largest model involved approximately one million neurons.

*Cochlear and neural filtering.* Head-filtered sounds were passed through a bank of fourth-order gammatone filters with center frequencies distributed on the ERB scale (central frequencies from 150 Hz to 5 kHz), modeling cochlear filtering (Glasberg and Moore, 1990). Linear filtering was carried out in parallel with a custom algorithm designed for large filterbanks (around 30,000 filters in our simulations). Gains and delays were then applied, with delays at most 1 ms and gains at most $\pm 10$ dB.

*Neuron model.* The filtered sounds were half-wave rectified and compressed by a $1/3$ power law $I = k([x]^+)^{1/3}$ (where $x$ is the sound pressure in pascals). The resulting signal was used as an input current to a leaky integrate-and-fire neuron with noise. The membrane potential $V$ evolves according to the equation:

$$\tau_m \frac{\mathrm{d}V}{\mathrm{d}t} = V_0 - V + I(t) + \sigma\sqrt{2\tau_m}\xi(t)$$

Table 1: Neuron model parameters

| Parameter | Value | Description |
|---|---|---|
| $V_r$ | -60 mV | Reset potential |
| $V_0$ | -60 mV | Resting potential |
| $V_t$ | -50 mV | Threshold potential |
| $t_{\mathrm{refrac}}$ | 5 ms | Absolute refractory period |
| | 0 ms | (for binaural neurons) |
| $\sigma$ | 1 mV | Standard deviation of membrane potential due to noise |
| $\tau_m$ | 1 ms | Membrane time constant |
| $W$ | 5 mV | Synaptic weight for coincidence detectors |
| $k$ | 0.2 V/Pa$^{1/3}$ | Acoustic scaling constant |

where $\tau_m$ is the membrane time constant, $V_0$ is the resting potential, $\xi(t)$ is Gaussian noise (such that $\langle \xi(t), \xi(s) \rangle = \delta(t-s)$) and $\sigma$ is the standard deviation of the membrane potential in the absence of spikes. When $V$ crosses the threshold $V_t$ a spike is emitted and $V$ is reset to $V_r$ and held there for an absolute refractory period $t_{\mathrm{refrac}}$. These neurons make synaptic connections with binaural neurons in a second layer (two presynaptic neurons for each binaural neuron). These coincidence detector neurons are leaky integrate-and-fire neurons with the same equations but their inputs are synaptic. Spikes arriving at these neurons cause an instantaneous increase $W$ in $V$ (where $W$ is the synaptic weight). Parameter values are given in Table 1.

*Estimating location from neural activation.* Each location is assigned an assembly of coincidence detector neurons, one in each frequency channel. When a sound is presented to the model, the total firing rate of all neurons in each assembly is computed. The estimated location is the one assigned to the maximally activated assembly. Figure 2 shows the activation of all location-specific assemblies in an example where a sound was presented to the model, after learning.

*Computing assemblies from HRTFs.* In the hardwired model, we defined the location-specific assemblies from the knowledge of HRTFs (the learning algorithm is explained in section 2.4). For a given location (filter pair $(F_L, F_R)$) and frequency channel (gammatone filter $G$), we choose the binaural neuron for which the the gains $(g_L, g_R)$ and delays $(d_L, d_R)$ of the two presynaptic monaural neurons minimize the RMS difference

$$\Delta = \sqrt{\int (g_L(G * F_L)(t - d_L) - g_R(G * F_R)(t - d_R))^2 \mathrm{d}t},$$

that is, the RMS difference between the inputs of the two neurons for a sound impulse at that location. We also impose $\max(g_L, g_R) = 1$ and $\max(d_L, d_R) = 0$ (so that one delay is null and the other is positive). The RMS difference is minimized when the delays correspond to the maximum of the cross-correlation between $L$ and $R$, $C(s) = \int (G * F_L)(t) \cdot (G * F_R)(t + s) \mathrm{d}t$, so that $C(d_R - d_L)$ is the maximum, and $g_R/g_L = C(d_R - d_L) / \int R(t)^2 \mathrm{d}t$.

## 2.4 Learning

In the hardwired model, the knowledge of the full set of HRTFs is used to estimate source location. But HRTFs are never directly accessible to the auditory system, because they are always convolved with the source signal. They cannot be genetically wired either, because they depend on the geometry of the head (which changes during development). In our model, when HRTFs are not explicitly known, location-specific assemblies are learned by presenting unknown sounds at different locations to the model, where there is one coincidence detector neuron for each choice of frequency, relative delay and relative gain. Relative delays were uniformly chosen between $-0.8$ ms and $0.8$ ms, and relative gains between $-8$ dB and $8$ dB uniformly on a dB scale. In total 69 relative delays were chosen and 61 relative gains. With 80 frequency channels, this gives a total of roughly $10^6$ neurons in the model. When a sound is presented at a given location, we define the assembly for this location by picking the maximally activated neuron in each frequency channel, as would be expected from a supervised Hebbian learning process with a teacher signal (e.g. visual cues). For practical reasons,

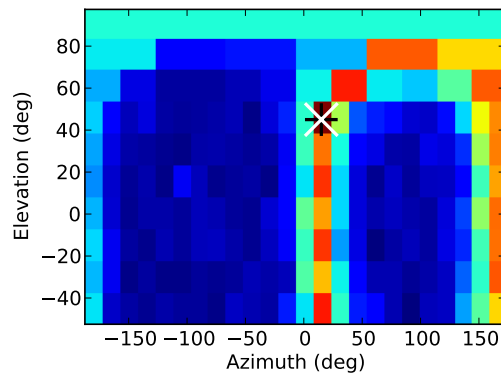

Figure 2: Activation of all location-specific assemblies in response to a sound coming from a particular location indicated by a black +. The white x shows the model estimate (maximally activated assembly). The mapping from assemblies to locations were learned from a set of sounds.

we did not implement this supervised learning with spiking models, but supervised learning with spiking neurons has been described in several previous studies (Song and Abbott, 2001; Davison and Frgnac, 2006; Witten et al., 2008).

## 3  Results

When the model is "hardwired" using the explicit knowledge of HRTFs, it can accurately localise a wide range of sounds (Figure 3A-C): for the maximum number of channels we tested (80), we obtained an average error of between 2 and 8 degrees for azimuth and 5 to 20 degrees for elevation (depending on sound type), and with more channels this error is likely to further decrease, as it did not appear to have reached an asymptote at 80 channels. Performance is better for sounds with broader spectrums, as each channel provides additional information. The model was also able to distinguish between sounds coming from the left and right (with an accuracy of almost 100%), and performed well for the more difficult tasks of distinguishing between front and back (80-85%) and between up and down (70-90%).

Figure 3D-F show the results using the learned best delays and gains, using the full training data set (seven sounds presented at each location, each of one second duration) and different test sounds. Performance is comparable to the hardwired model. Average azimuth errors for 80 channels are 4-8 degrees, and elevation errors are 10-27 degrees. Distinguishing left and right is done with close to 100% accuracy, front and back with 75-85% and up and down with 65-90%. Figure 4 shows how the localisation accuracy improves with more training data. With only a single sound of one second duration at each location, the performance is already very good. Increasing the training to three seconds of training data at each location improves the accuracy, but including further training data does not appear to lead to any significant improvement. Although it is close, the performance does not seem to converge to that of the hardwired model, which might be due to a limited sampling of delays and gains (69 relative delays and 61 relative gains), or perhaps to the presence of physiological noise in our models (Goodman and Brette, in press).

Figure 5 shows the properties of neurons in a location-specific assembly: interaural delay (Figure 5A) and interaural gain difference (Figure 5B) for each frequency channel. For this location, the assemblies in the hardwired model and with learning were very similar, which indicates that the learning procedure was indeed able to catch the binaural cues associated with that location. The distributions of delays and gain differences were similar in the hardwired model and with learning. In the hardwired model, these interaural delays and gains correspond to the ITDs and ILDs in fine frequency bands. To each location corresponds a specific frequency-dependent pattern of ITDs and ILDs, which is informative of both azimuth and elevation. In particular, these patterns are different when the location is reversed between front and back (not shown), and this difference is exploited by the model to distinguish between these two cases.

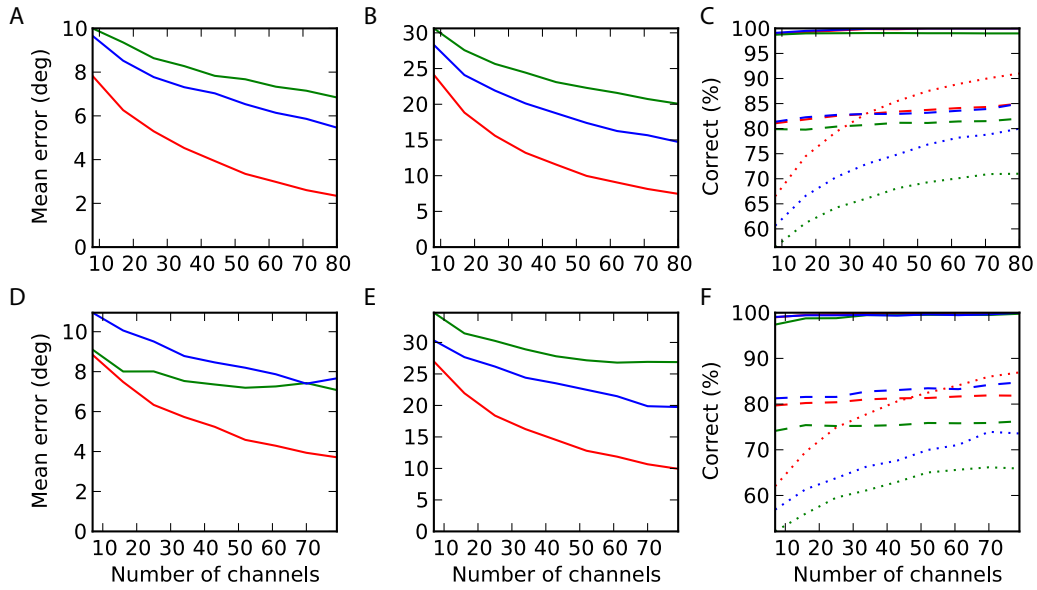

Figure 3: Performance of the hard-wired model (A-C) and with learning (D-F). A, D, Mean error in azimuth estimates as a function of the number of frequency channels (i.e., assembly size) for white noise (red), vowel-consonant-vowel (blue) and musical instruments (green). Front-back reversed locations were considered as having the same azimuth. The channels were selected at random between 150 Hz and 5 kHz and results were averaged over many random choices. B, E, Mean error in elevation estimates. C, F, Categorization performance discriminating left and right (solid), front and back (dashed) and up and down (dotted).

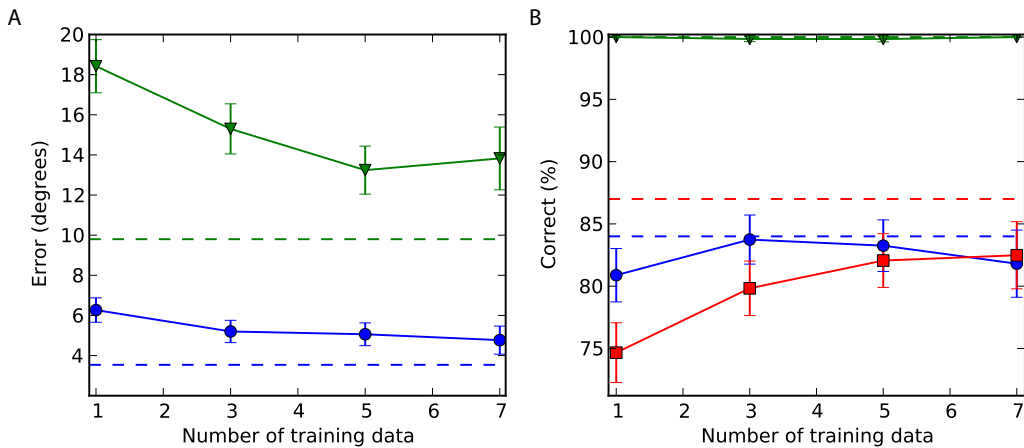

Figure 4: Performance improvement with training (80 frequency channels). A, Average estimation error in azimuth (blue) and elevation (green) as a function of the number of sounds presented at each location during learning (each sound lasts 1 second). The error bars represent 95% confidence intervals. The dashed lines indicate the estimation error in the hardwired model (when HRTFs are explicitly known). B, Categorization performance vs. number of sounds per location for discriminating left and right (green), front and back (blue) and up and down (red).

## 4   Discussion

The sound produced by a source propagates to the ears according to linear laws. Thus the ears receive two differently filtered versions of the same signal, which induce a location-specific structure

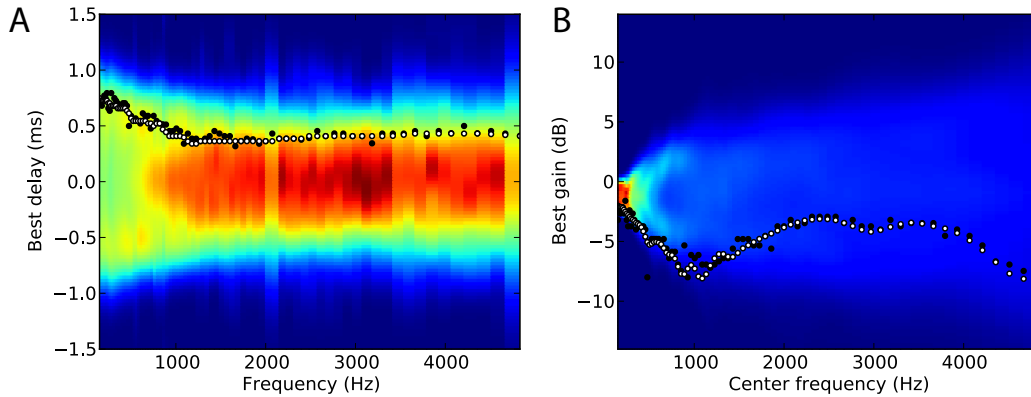

Figure 5: Location-specific assembly in the hardwired model and with learning. A, Preferred interaural delay vs. preferred frequency for neurons in an assembly corresponding to one particular location, in the hardwired model (white circles) and with learning (black circles). The colored background shows the distribution of preferred delays in all neurons in the hardwired model. B, Interaural gain difference vs. preferred frequency for the same assemblies.

in the binaural stimulus. When binaural signals are transformed by a heterogeneous population of neurons, this structure is mapped to synchrony patterns, which are location-specific. We designed a simple spiking neuron model which exploits this property to estimate the location of sound sources in a way that is independent of the source signal. In the model, each location activates a specific assembly. We showed that the mapping between assemblies and locations can be directly learned in a supervised way from the presentation of a set of sounds at different locations, with no previous knowledge of the HRTFs or the sounds. With 80 frequency channels, we found that 1 second of training data per location was enough to estimate the azimuth of a new sound with mean error 6 degrees and the elevation with error 18 degrees.

Humans can learn to localise sound sources when their acoustical cues change, for example when molds are inserted into their ears (Hofman et al., 1998; Zahorik et al., 2006). Learning a new mapping can take a long time (several weeks in the first study), which is consistent with the idea that the new mapping is learned from exposure to sounds from known locations. Interestingly, the previous mapping is instantly recovered when the ear molds are removed, meaning that the representations of the two acoustical environments do not interfere. This is consistent with our model, in which two acoustical environments would be represented by two possibly overlapping sets of neural assemblies.

In our model, we assumed that the receptive field of monaural neurons can be modeled as a band-pass filter with various gains and delays. Differences in input gains could simply arise from differences in membrane resistance, or in the number and strength of the synapses made by auditory nerve fibers. Delays could arise from many causes: axonal delays (either presynaptic or postsynaptic), cochlear delays (Joris et al., 2006), inhibitory delays (Brand et al., 2002). The distribution of best delays of the binaural neurons in our model reflect the distribution of ITDs in the acoustical environment. This contradicts the observation in many species that the best delays are always smaller than half the characteristic period, i.e., they are within the $\pi$-limit (Joris and Yin, 2007). However, we checked that the model performed almost equally well with this constraint (Goodman and Brette, in press), which is not very surprising since best delays above the $\pi$-limit are mostly redundant. In small mammals (guinea pigs, gerbils), it has been shown that the best phases of binaural neurons in the MSO and IC are in fact even more constrained, since they are scattered around $\pm\pi/4$, in constrast with birds (e.g. barn owl) where the best phases are continuously distributed (Wagner et al., 2007). However, in larger mammals such as cats, best IPDs in the MSO are more continuously distributed (Yin and Chan, 1990), with a larger proportion close to 0 (Figure 18 in Yin and Chan, 1990). It has not been measured in humans, but the same optimal coding theory that predicts the discrete distribution of phases in small mammals predicts that best delays should be continuously distributed above 400 Hz (80% of the frequency channels in our model). In addition, psychophysical results also

imply that humans can estimate both the azimuth and elevation of low-pass filtered sound sources ($< 3$ kHz) (Algazi et al., 2001), which only contain binaural cues. This is contradictory with the two-channel model (best delays at $\pm\pi/4$) and in agreement with ours (including the fact that elevation could only be estimated away from the median plane in these experiments).

Our model is conceptually similar to a recent signal processing method (with no neural implementation) to localize sound sources in the horizontal plane (Macdonald, 2008), where coincidence detection is replaced by Pearson correlation between the two transformed monaural broadband signals (no filterbank). However, that method requires explicit knowledge of the HRTFs, so that it cannot be directly learned from natural exposure to sounds.

The HRTFs used in our virtual acoustic environment were recorded at a constant distance, so that we could only test the model performance in estimating the azimuth and elevation of a sound source. However, in principle, it should also be able to estimate the distance when the source is close. It should also apply equally well to non-anechoic environments, because our model only relies on the linearity of sound propagation. However, a difficult task, which we have not addressed, is to locate sounds in a new environment, because reflections would change the binaural cues and therefore the location-specific assemblies. One possibility would be to isolate the direct sound from the reflections, but this requires additional mechanisms, which probably underlie the precedence effect (Litovsky et al., 1999).

# References

Algazi, V. R., C. Avendano, and R. O. Duda (2001, March). Elevation localization and head-related transfer function analysis at low frequencies. *The Journal of the Acoustical Society of America 109*(3), 1110–1122.

Brand, A., O. Behrend, T. Marquardt, D. McAlpine, and B. Grothe (2002). Precise inhibition is essential for microsecond interaural time difference coding. *Nature 417*(6888), 543.

Colburn, H. S. (1973, December). Theory of binaural interaction based on auditory-nerve data. i. general strategy and preliminary results on interaural discrimination. *The Journal of the Acoustical Society of America 54*(6), 1458–1470.

Davison, A. P. and Y. Frgnac (2006, May). Learning Cross-Modal spatial transformations through spike Timing-Dependent plasticity. *J. Neurosci. 26*(21), 5604–5615.

Gaik, W. (1993, July). Combined evaluation of interaural time and intensity differences: Psychoacoustic results and computer modeling. *The Journal of the Acoustical Society of America 94*(1), 98–110.

Gerstner, W., R. Kempter, J. L. van Hemmen, and H. Wagner (1996). A neuronal learning rule for sub-millisecond temporal coding. *Nature 383*(6595), 76.

Glasberg, B. R. and B. C. Moore (1990, August). Derivation of auditory filter shapes from notched-noise data. *Hearing Research 47*(1-2), 103–138. PMID: 2228789.

Goodman, D. F. M. and R. Brette (2009). The Brian simulator. *Frontiers in Neuroscience 3*(2), 192–197.

Goodman, D. F. M. and R. Brette (in press). Spike-timing-based computation in sound localization. *PLoS Comp. Biol.*.

Harper, N. S. and D. McAlpine (2004). Optimal neural population coding of an auditory spatial cue. *Nature 430*(7000), 682–686.

Hofman, P. M., J. G. V. Riswick, and A. J. V. Opstal (1998). Relearning sound localization with new ears. *Nat Neurosci 1*(5), 417–421.

Jeffress, L. A. (1948, February). A place theory of sound localization. *Journal of Comparative and Physiological Psychology 41*(1), 35–9. PMID: 18904764.

Joris, P. and T. C. T. Yin (2007, February). A matter of time: internal delays in binaural processing. *Trends in Neurosciences 30*(2), 70–8. PMID: 17188761.

Joris, P. X., B. V. de Sande, D. H. Louage, and M. van der Heijden (2006). Binaural and cochlear disparities. *Proceedings of the National Academy of Sciences 103*(34), 12917.

Lindemann, W. (1986, December). Extension of a binaural cross-correlation model by contralateral inhibition. i. simulation of lateralization for stationary signals. *The Journal of the Acoustical Society of America 80*(6), 1608–1622.

Litovsky, R. Y., H. S. Colburn, W. A. Yost, and S. J. Guzman (1999, October). The precedence effect. *The Journal of the Acoustical Society of America 106*(4), 1633–1654.

Liu, J., H. Erwin, S. Wermter, and M. Elsaid (2008). A biologically inspired spiking neural network for sound localisation by the inferior colliculus. In *Artificial Neural Networks - ICANN 2008*, pp. 396–405.

Lorenzi, C., F. Berthommier, F. Apoux, and N. Bacri (1999, October). Effects of envelope expansion on speech recognition. *Hearing Research 136*(1-2), 131–138.

Macdonald, J. A. (2008, June). A localization algorithm based on head-related transfer functions. *The Journal of the Acoustical Society of America 123*(6), 4290–4296. PMID: 18537380.

Reed, M. C. and J. J. Blum (1990, September). A model for the computation and encoding of azimuthal information by the lateral superior olive. *The Journal of the Acoustical Society of America 88*(3), 1442–1453. PMID: 2229677.

Song, S. and L. F. Abbott (2001, October). Cortical development and remapping through spike Timing-Dependent plasticity. *Neuron 32*(2), 339–350.

Wagner, H., A. Asadollahi, P. Bremen, F. Endler, K. Vonderschen, and M. von Campenhausen (2007). Distribution of interaural time difference in the barn owl's inferior colliculus in the low- and High-Frequency ranges. *J. Neurosci. 27*(15), 4191–4200.

Witten, I. B., E. I. Knudsen, and H. Sompolinsky (2008, August). A hebbian learning rule mediates asymmetric plasticity in aligning sensory representations. *J Neurophysiol 100*(2), 1067–1079.

Yin, T. C. and J. C. Chan (1990). Interaural time sensitivity in medial superior olive of cat. *J Neurophysiol 64*(2), 465–488.

Zahorik, P., P. Bangayan, V. Sundareswaran, K. Wang, and C. Tam (2006, July). Perceptual recalibration in human sound localization: Learning to remediate front-back reversals. *The Journal of the Acoustical Society of America 120*(1), 343–359.

Zhou, Y., L. H. Carney, and H. S. Colburn (2005, March). A model for interaural time difference sensitivity in the medial superior olive: Interaction of excitatory and inhibitory synaptic inputs, channel dynamics, and cellular morphology. *J. Neurosci. 25*(12), 3046–3058.

